# AN OPTIMALITY PRINCIPLE FOR UNSUPERVISED LEARNING

Terence D. Sanger
MIT AI Laboratory, NE43-743
Cambridge, MA 02139
(tds@wheaties.ai.mit.edu)

## ABSTRACT

We propose an optimality principle for training an unsupervised feedforward neural network based upon maximal ability to reconstruct the input data from the network outputs. We describe an algorithm which can be used to train either linear or nonlinear networks with certain types of nonlinearity. Examples of applications to the problems of image coding, feature detection, and analysis of random-dot stereograms are presented.

## 1. INTRODUCTION

There are many algorithms for unsupervised training of neural networks, each of which has a particular optimality criterion as its goal. (For a partial review, see (Hinton, 1987, Lippmann, 1987).) We have presented a new algorithm for training single-layer linear networks which has been shown to have optimality properties associated with the Karhunen-Loève expansion (Sanger, 1988b). We now show that a similar algorithm can be applied to certain types of nonlinear feedforward networks, and we give some examples of its behavior.

The optimality principle which we will use to describe the algorithm is based on the idea of maximizing information which was first proposed as a desirable property of neural networks by Linsker (1986, 1988). Unfortunately, measuring the information in network outputs can be difficult without precise knowledge of the distribution on the input data, so we seek another measure which is related to information but which is easier to compute. If instead of maximizing information, we try to maximize our ability to reconstruct the input (with minimum mean-squared error) given the output of the network, we are able to obtain some useful results. Note that this is not equivalent to maximizing information except in some special cases. However, it contains the intuitive notion that the input data is being represented by the network in such a way that very little of it has been "lost".

# 2. LINEAR CASE

We now summarize some of the results in (Sanger, 1988b). A single-layer linear feedforward network is described by an $M \times N$ matrix $C$ of weights such that if $x$ is a vector of $N$ inputs and $y$ is a vector of $M$ outputs with $M < N$, we have $y = Cx$. As mentioned above, we choose an optimality principle defined so that we can best reconstruct the inputs to the network given the outputs. We want to minimize the mean squared error $E[(x - \hat{x})^2]$ where $x$ is the actual input which is zero-mean with correlation matrix $Q = E[xx^T]$, and $\hat{x}$ is a linear estimation of this input given the output $y$. The linear least squares estimate (LLSE) is given by

$$\hat{x} = QC^T(CQC^T)^{-1}y$$

and we will assume that $\hat{x}$ is computed in this way for any matrix $C$ of weights which we choose. The mean-squared error for the LLSE is given by

$$E[(x - \hat{x})^2] = Q - QC^T(CQC^T)^{-1}CQ$$

and it is well known that this is minimized if the rows of $C$ are a linear combination of the first $M$ eigenvectors of the correlation matrix $Q$. One optimal choice of $C$ is the Singular Value Decomposition (SVD) of $Q$, for which the output correlation matrix $E[yy^T] = CQC^T$ will be the diagonal matrix of eigenvalues of $Q$. In this case, the outputs are uncorrelated and the sum of their variances (trace $E[yy^T]$) is maximal for any set of $M$ uncorrelated outputs. We can thus think of the eigenvectors as being obtained by any process which maximizes the output variance while maintaining the outputs uncorrelated.

We now define the optimal single-layer linear network as that network whose weights represent the first $M$ eigenvectors of the input correlation matrix $Q$. The optimal network thus minimizes the mean-squared approximation error $E[(x - \hat{x})^2]$ given the shape constraint that $M < N$.

## 2.1 LINEAR ALGORITHM

We have previously proposed a weight-update rule called the "Generalized Hebbian Algorithm", and proven that this algorithm causes the rows of the weight matrix $C$ to converge to the eigenvectors of the input correlation matrix $Q$ (Sanger, 1988a,b). The algorithm is given by:

$$C(t + 1) = C(t) + \gamma \left( y(t)x^T(t) - \text{LT}[y(t)y^T(t)]C(t) \right) \tag{1}$$

where $\gamma$ is a rate constant which decreases as $1/t$, $x(t)$ is an input sample vector, $y(t) = C(t)x(t)$, and LT[] is an operator which makes its matrix argument lower triangular by setting all entries above the diagonal to zero. This algorithm can be implemented using only a local synaptic learning rule (Sanger, 1988b).

Since the Generalized Hebbian Algorithm computes the eigenvectors of the input correlation matrix $Q$, it is related to the Singular Value Decomposition (SVD),

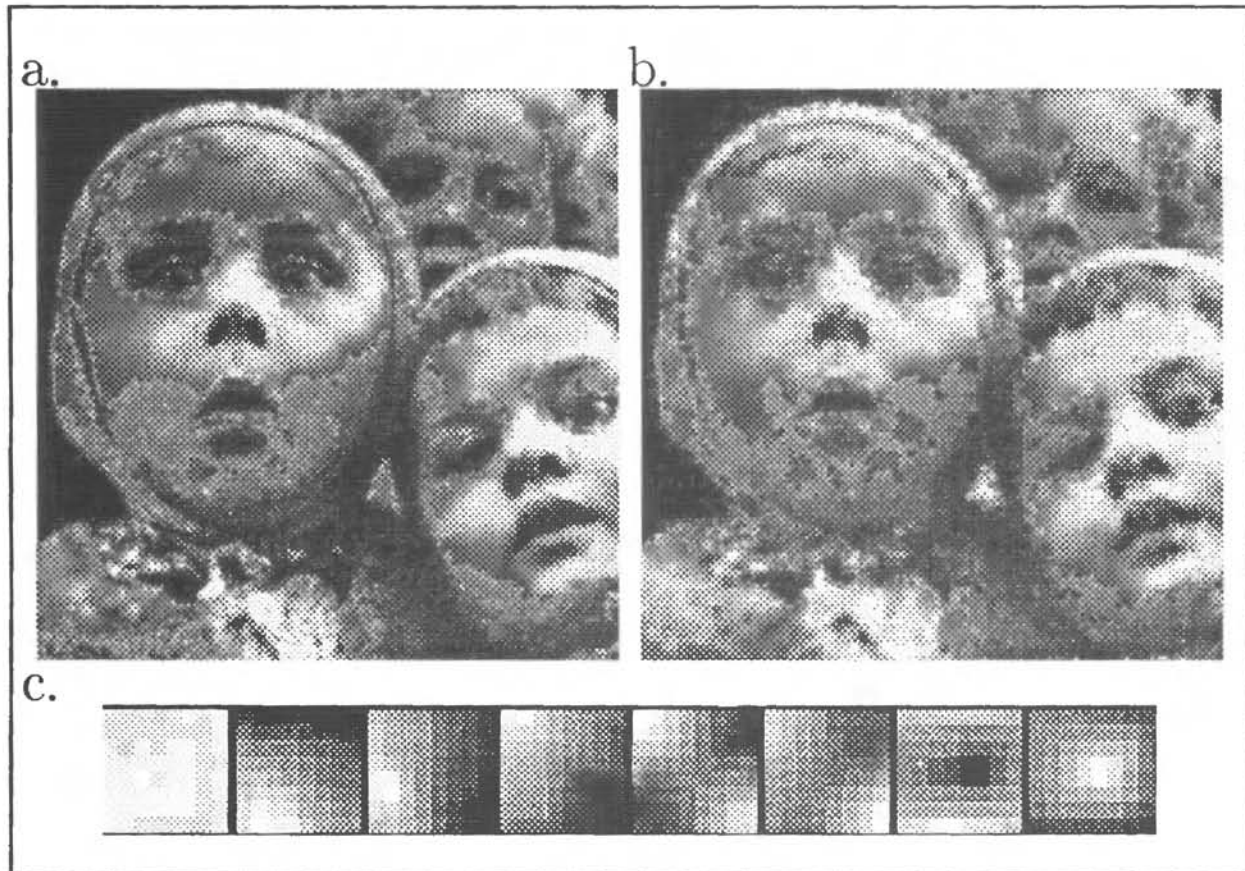

*Figure 1: (a) original image. (b) image coded at .36 bits per pixel. (c) masks learned by the network which were used for vector quantized coding of 8x8 blocks of the image.*

Principal Components Analysis (PCA), and the Karhunen-Loève Transform (KLT). (For a review of several related algorithms for performing the KLT, see (Oja, 1983).)

## 2.2 IMAGE CODING

We present one example of the behavior of a single-layer linear network. (This example appears in (Sanger, 1988b).) Figure 1a shows an original 256x256x8bit image which was used for training a network. 8x8 blocks of the image were chosen by scanning over the image, and these were used as training inputs to a network with 64 inputs and 8 outputs. After training, the set of weights for each output (figure 1c) represents a vector quantizing mask. Each 8x8 block of the input image is then coded using the outputs of the network. Each output is quantized with a number of bits related to the log of the variance, and the original figure is approximated from the quantized outputs. The reconstruction of figure 1b uses a total of 23 bits per 8x8 region, which gives a data rate of 0.36 bits per pixel. The fact that the image could be represented using such a low bit rate indicates that the masks that were found represent significant features which are useful for recognition. This image coding technique is equivalent to block-coded KLT methods common in the literature.

# 3. NONLINEAR CASE

In general, training a nonlinear unsupervised network to approximate nonlinear functions is very difficult. Because of the large (infinite-dimensional) space of possible functions, it is important to have detailed knowledge of the class of functions which are useful in order to design an efficient network algorithm. (Several people pointed out to me that the talk implied such knowledge is not necessary, but unfortunately such an implication is false.)

The network structure we consider is a linear layer represented by a matrix $C$ (which is perhaps an interior layer of a larger network) followed by node nonlinearities $\sigma(y_i)$ where $y_i$ is the $i^{th}$ linear output, followed by another linear layer (perhaps followed by more layers). We assume that the nonlinearities $\sigma()$ are fixed, and that the only parameters susceptible to training are the linear weights $C$.

If $z$ is the $M$-vector of outputs after the nonlinearity, then we can write each component $z_i = \sigma(y_i) = \sigma(c_i x)$ where $c_i$ is the $i^{th}$ row of the matrix $C$. Note that the level contours of each function $z_i$ are determined entirely by the vector $c_i$, and that the effect of $\sigma()$ is limited to modifying the output value. Intuitively, we thus expect that if $y_i$ encodes a useful parameter of the input $x$, then $z_i$ will encode the same parameter, although scaled by the nonlinearity $\sigma()$.

This can be formalized, and if we choose our optimality principle to again be minimum mean-squared linear approximation of the original input $x$ given the output $z$, the best solution remains when the rows of $C$ are a linear combination of the first $M$ eigenvectors of the input correlation matrix $Q$ (Bourlard and Kamp, 1988).

In two of the simulations, the nonlinearity $\sigma()$ which we use is a rectification nonlinearity, given by

$$\sigma(y_i) = \begin{cases} y_i & \text{if } y_i \geq 0 \\ 0 & \text{if } y_i < 0 \end{cases}$$

Note that at most one of $\{\sigma(y_i), \sigma(-y_i)\}$ is nonzero at any time, so these two values are uncorrelated. Therefore, if we maximize the variance of $y$ (before the nonlinearity) while maintaining the elements of $z$ (after the nonlinearity) uncorrelated, we need $2M$ outputs in order to represent the data available from an $M$-vector $y$. Note that $2M$ may be greater than the number of inputs $N$, so that the "hidden layer" $z$ can have more elements than the input.

## 3.1 NONLINEAR ALGORITHM

The nonlinear Generalized Hebbian Algorithm has exactly the same form as for the linear case, except that we substitute the use of the output values *after* the nonlinearity for the linear values. The algorithm is thus given by:

$$C(t+1) = C(t) + \gamma \left( z(t)x^T(t) - LT[z(t)z^T(t)]C(t) \right) \tag{2}$$

where the elements of $z$ are given by $z_i(t) = \sigma(y_i(t))$, with $y(t) = C(t)x(t)$.

Although we have not proven that this algorithm converges, a heuristic analysis of its behavior (for a rectification nonlinearity and Gaussian input distribution)

shows that stable points may exist for which each row of $C$ is proportional to an eigenvector of $Q$, and pairs of rows are either the negative of each other or orthogonal. In practice, the rows of $C$ are ordered by decreasing output variance, and occur in pairs for which one member is the negative of the other. This choice of $C$ will maximize the sum of the output variances for uncorrelated outputs, so long as the input is Gaussian. It also allows optimal linear estimation of the input given the output, so long as both polarities of each of the eigenvectors are present.

## 3.2 NONLINEAR EXAMPLES

### 3.2.1 Encoder Problem

We compare the performance of two nonlinear networks which have learned to perform an identity mapping (the "encoder" problem). One is trained by backpropagation, (Rumelhart *et al.*, 1986) and the other has two hidden layers trained using the unsupervised Hebbian algorithm, while the output layer is trained using a supervised LMS algorithm (Widrow and Hoff, 1960). The network has 5 inputs, two hidden layers of 3 units each, and 5 outputs. There is a sigmoid nonlinearity at each hidden layer, but the thresholds are all kept at zero. The task is to minimize the mean-squared difference between the inputs and the outputs. The input is a zero-mean correlated Gaussian random 5-vector, and both algorithms are presented with the same sequence of inputs. The unsupervised-trained network converged to a steady state after 1600 examples, and the backpropagation network converged after 2400 (convergence determined by no further decrease in average error). The RMS error at steady state was 0.42 for both algorithms (this figure should be compared to the sum of the variances of the inputs, which was 5.0). Therefore, for this particular task, there is no significant difference in performance between backpropagation and the Generalized Hebbian Algorithm. This is an encouraging result, since if we can use an unsupervised algorithm to solve other problems, the training time will scale at most linearly with the number of layers.

### 3.2.2 Nonlinear Receptive Fields

Several investigators have shown that Hebbian algorithms can discover useful image features related to the receptive fields of cells in primate visual cortex (see for example (Bienenstock *et al.*, 1982, Linsker, 1986, Barrow, 1987)). One of the more recent methods uses an algorithm very similar to the one proposed here to find the principal component of the input (Linsker, 1986). We performed an experiment to find out what types of nonlinear receptive fields could be learned by the Generalized Hebbian Algorithm if provided with similar input to that used by Linsker.

We used a single-layer nonlinear network with 4096 inputs arranged in a 64x64 grid, and 16 outputs with a rectification nonlinearity. The input data consisted of images of low-pass filtered white Gaussian noise multiplied by a Gaussian window. After 5000 samples, the 16 outputs learned the masks shown in figure 2. These masks possess qualitative similarity to the receptive fields of cells found in the visual cortex of cat and monkey (see for example (Andrews and Pollen, 1979)). They are equivalent to the masks learned by a purely linear network (Sanger, 1988b), except that both positive and negative polarities of most mask shapes are present here.

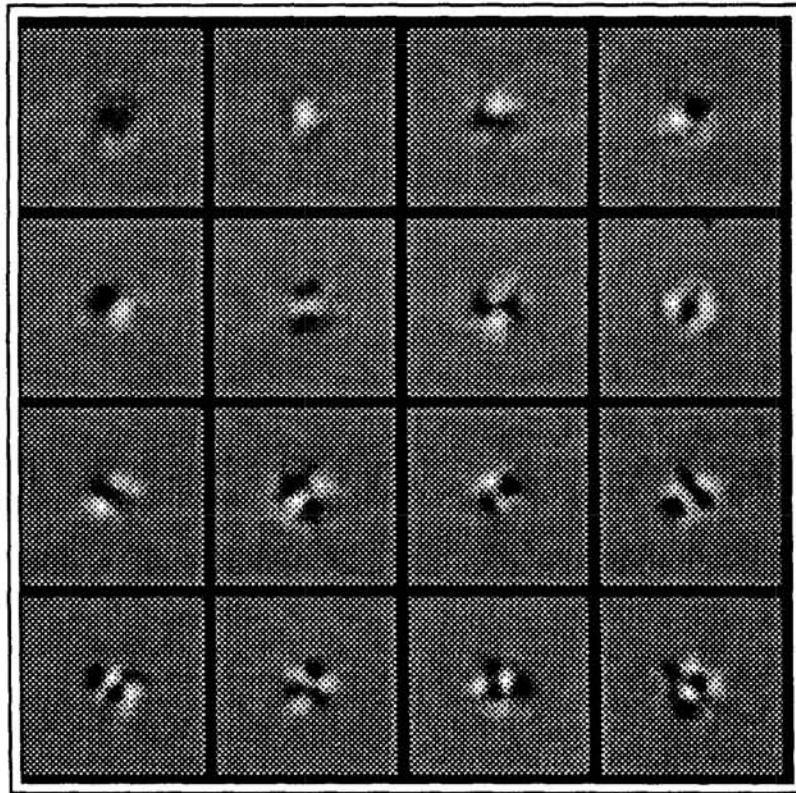

*Figure 2: Nonlinear receptive fields ordered from left-to-right and top-to-bottom.*

### 3.2.3 Stereo

We now show how the nonlinear Generalized Hebbian Algorithm can be used to train a two-layer network to detect disparity edges. The network has 128 inputs, 8 types of unit in the hidden layer with a rectification nonlinearity, and 4 types of output unit. A 128x128 pixel random-dot stereo pair was generated in which the left half had a disparity of two pixels, and the right half had zero disparity. The image was convolved with a vertically-oriented elliptical Gaussian mask to remove high-frequency vertical components. Corresponding 8x8 blocks of the left and right images (64 pixels from each image) were multiplied by a Gaussian window function and presented as input to the network, which was allowed to learn the first layer according to the unsupervised algorithm. After 4000 iterations, the first layer had converged to a set of 8 pairs of masks. These masks were convolved with the images (the left mask was convolved with the left image, and the right mask with the right image, and the two results were summed and rectified) to produce a pattern of activity at the hidden layer. (Although there were only 8 types of hidden unit, we now allow one of each type to be centered at every input image location to obtain a pattern of total activity.) Figure 3 shows this activity, and we can see that the last four masks are disparity-sensitive since they respond preferentially to either the 2 pixel disparity or the zero disparity region of the image.

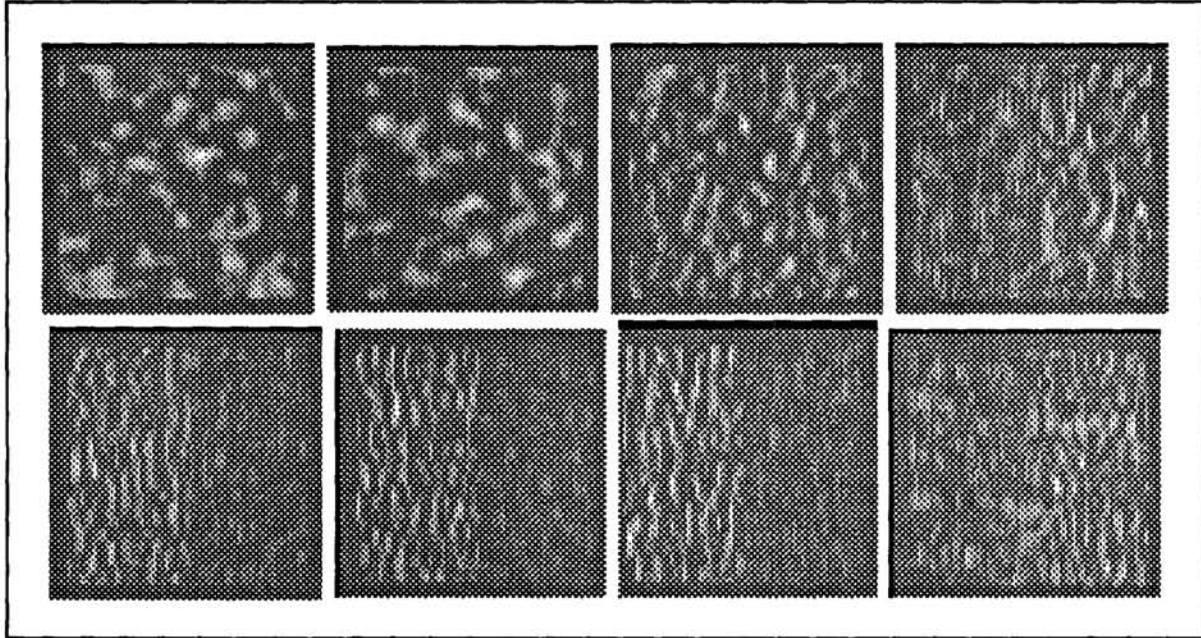

*Figure 3: Hidden layer response for a two-layer nonlinear network trained on stereo images. The left half of the input random dot image has a 2 pixel disparity, and the right half has zero disparity.*

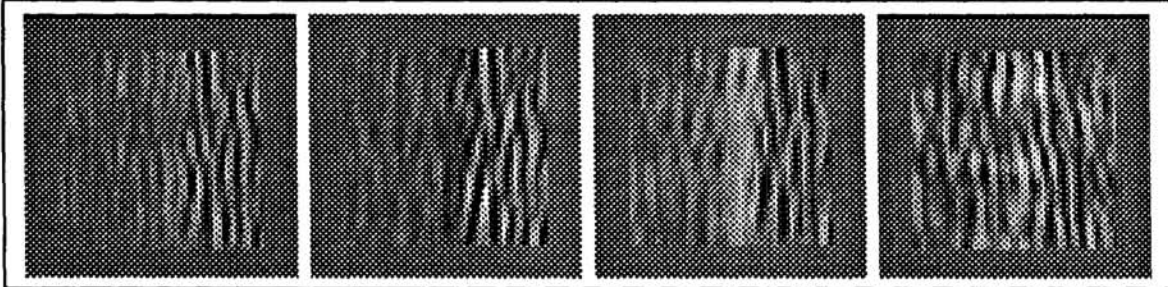

*Figure 4: Output layer response for a two-layer nonlinear network trained on stereo images.*

Since we were interested in disparity, we trained the second layer using only the last four hidden unit types. The second layer had 1024 (=4x16x16) inputs organized as a 16x16 receptive field in each of the four hidden unit "planes". The outputs did not have any nonlinearity. Training was performed by scanning over the hidden unit activity pattern (successive examples overlapped by 8 pixels) and 6000 iterations were used to produce the second-layer weights. The masks that were learned were then convolved with the hidden unit activity pattern to produce an output unit activity pattern, shown in figure 4.

The third output is clearly sensitive to a change in disparity (a depth edge). If we generate several different random-dot stereograms and average the output results,

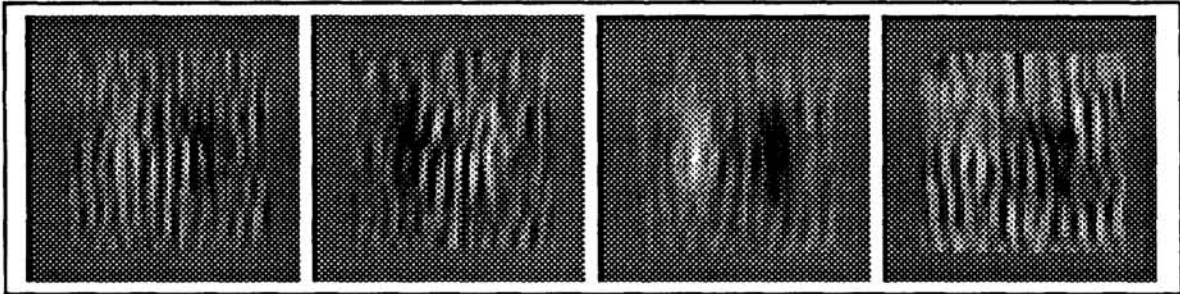

*Figure 5: Output layer response averaged over ten stereograms with a central 2 pixel disparity square and zero disparity surround.*

we see that the other outputs are also sensitive (on average) to disparity changes, but not as much as the third. Figure 5 shows the averaged response to 10 stereograms with a central 2 pixel disparity square against a zero disparity background. Note that the ability to detect disparity edges requires the rectification nonlinearity at the hidden layer, since no linear function has this property.

## 4. CONCLUSION

We have shown that the unsupervised Generalized Hebbian Algorithm can produce useful networks. The algorithm has been proven to converge only for single-layer linear networks. However, when applied to nonlinear networks with certain types of nonlinearity, it appears to converge to good results. In certain cases, it operates by maintaining the outputs uncorrelated while maximizing their variance. We have not investigated its behavior on nonlinearities other than rectification or sigmoids, so we can make no predictions about its general utility. Nevertheless, the few examples presented for the nonlinear case are encouraging, and suggest that further investigation of this algorithm will yield interesting results.

**Acknowledgements**

I would like to express my gratitude to the many people at the NIPS conference and elsewhere whose comments, criticisms, and suggestions have increased my understanding of these results. In particular, thanks are due to Ralph Linsker for pointing out to me an important error in the presentation and for his comments on the manuscript, as well as to John Denker, Steve Nowlan, Rich Sutton, Tom Breuel, and my advisor Tomaso Poggio.

This report describes research done at the MIT Artificial Intelligence Laboratory, and sponsored by a grant from the Office of Naval Research (ONR), Cognitive and Neural Sciences Division; by the Alfred P. Sloan Foundation; by the National Science Foundation; by the Artificial Intelligence Center of Hughes Aircraft Corporation (S1-801534-2); and by the NATO Scientific Affairs Division (0403/87). Support for the A. I. Laboratory's artificial intelligence research is provided by the

Advanced Research Projects Agency of the Department of Defense under Army contract DACA76–85–C–0010, and in part by ONR contract N00014–85–K–0124. The author was supported during part of this research by a National Science Foundation Graduate fellowship, and later by a Medical Scientist Training Program grant.

## References

Andrews B. W., Pollen D. A., 1979, Relationship between spatial frequency selectivity and receptive field profile of simple cells, *J. Physiol.*, 287:163–176.

Barrow H. G., 1987, Learning receptive fields, In *Proc. IEEE 1st Ann. Conference on Neural Networks*, volume 4, pages 115–121, San Diego, CA.

Bienenstock E. L., Cooper L. N., Munro P. W., 1982, Theory for the development of neuron selectivity: Orientation specificity and binocular interaction in visual cortex, *J. Neuroscience*, 2(1):32–48.

Bourlard H., Kamp Y., 1988, Auto-association by multilayer perceptrons and singular value decomposition, *Biological Cybernetics*, 59:291–294.

Hinton G. E., 1987, Connectionist learning procedures, CMU Tech. Report CS-87-115.

Linsker R., 1986, From basic network principles to neural architecture, *Proc. Natl. Acad. Sci. USA*, 83:7508–7512.

Linsker R., 1988, Self-organization in a perceptual network, *Computer*, 21(3):105–117.

Lippmann R. P., 1987, An introduction to computing with neural nets, *IEEE ASSP Magazine*, pages 4–22.

Oja E., 1983, *Subspace Methods of Pattern Recognition*, Research Studies Press, UK.

Rumelhart D. E., Hinton G. E., Williams R. J., 1986, Learning representations by back-propagating errors, *Nature*, 323(9):533–536.

Sanger T. D., 1988a, Optimal unsupervised learning, *Neural Networks*, 1(S1):127, Proc. 1st Ann. INNS meeting, Boston, MA.

Sanger T. D., 1988b, Optimal unsupervised learning in a single-layer linear feedforward neural network, submitted to *Neural Networks*.

Widrow B., Hoff M. E., 1960, Adaptive switching circuits, In *IRE WESCON Conv. Record, Part 4*, pages 96–104.